# Linear concepts and hidden variables: An empirical study

**Adam J. Grove**
NEC Research Institute
4 Independence Way
Princeton NJ 08540
grove@research.nj.nec.com

**Dan Roth**[*]
Department of Computer Science
University of Illinois at Urbana-Champaign
1304 W. Springfield Ave. Urbana 61801
danr@cs.uiuc.edu

## Abstract

Some learning techniques for classification tasks work indirectly, by first trying to fit a full probabilistic model to the observed data. Whether this is a good idea or not depends on the robustness with respect to deviations from the postulated model. We study this question experimentally in a restricted, yet non-trivial and interesting case: we consider a *conditionally independent attribute* (CIA) model which postulates a single binary-valued *hidden variable* $z$ on which all other attributes (i.e., the target and the observables) depend. In this model, finding the most likely value of any one variable (given known values for the others) reduces to testing a linear function of the observed values.

We learn CIA with two techniques: the standard EM algorithm, and a new algorithm we develop based on covariances. We compare these, in a controlled fashion, against an algorithm (a version of *Winnow*) that attempts to find a good linear classifier directly. Our conclusions help delimit the fragility of using the CIA model for classification: once the data departs from this model, performance quickly degrades and drops below that of the directly-learned linear classifier.

## 1 Introduction

We consider the classic task of predicting a *binary* (0/1) target variable $x_0$, based on the values of some $n$ other binary variables $x_1 \ldots x_n$. We can distinguish between two styles of learning approach for such tasks. *Parametric* algorithms postulate some form of probabilistic model underlying the data, and try to fit the model's parameters. To classify an example we can compute the conditional probability distribution for $x_0$ given the values of the known variables, and then predict the most probable value. *Non-parametric* algorithms do not assume that the training data has a particular form. They instead search directly in the space of possible classification functions, attempting to find one with small error on the training set of examples.

An important advantage of parametric approaches is that the induced model can be used to support a wide range of inferences, aside from the specified classification task. On the other hand, to postulate a particular form of probabilistic model can be a very strong assumption.

---

[*]Partly supported by ONR grant N00014-96-1-0550 while visiting Harvard University.

So it is important to understand how robust such methods are when the real world deviates from the assumed model.

In this paper, we report on some experiments that test this issue. We consider the specific case of $n + 1$ *conditionally independent attributes* $x_i$ together with a single *unobserved* variable $z$, also assumed to be binary valued, on which the $x_i$ depend (henceforth, the binary CIA model); see Section 2. In fact, such models are plausible in many domains (for instance, in some language interpretation tasks; see [GR96]). We fit the parameters of the CIA model using the well-known *expectation-maximization* (EM) technique [DLR77], and also with a new algorithm we have developed based on estimating covariances; see Section 4. In the nonparametric case, we simply search for a good *linear separator.* This is because the optimal predictors for the binary CIA model (i.e., for predicting one variable given known values for the rest) are also linear. This means that our comparison is "fair" in the sense that neither strategy can choose from classifiers with more expressive power than the other. As a representative of the non-parametric class of algorithms, we use the Winnow algorithm of [Lit88], with some modifications (see Section 6). Winnow works directly to find a "good" linear separator. It is guaranteed to find a perfect separator if one exists, and empirically seems to be fairly successful even when there is no perfect separator [GR96, Blu97]. It is also very fast.

Our experimental methodology is to first generate synthetic data from a true CIA model and test performance; we then study various deviations from the model. There are various interesting issues involved in constructing good experiments, including the desirability of controlling the inherent "difficulty" of learning a model. Since we cannot characterize the entire space, we consider here only deviations in which the data is drawn from a CIA model in which the hidden variable can take more than two values. (Note that the optimal classifier given $x_0$ is generally not linear in this case.)

Our observations are not *qualitatively* surprising. CIA does well when the assumed model is correct, but performance degrades when the world departs from the model. But as we discuss, we found it surprising how fragile this model can sometimes be, when compared against algorithms such as Winnow. This is even though the data is not linearly separable either, and so one might expect the direct learning techniques to degrade in performance as well. But it seems that Winnow and related approaches are far less fragile. Thus the main contribution of this work is that our results shed light on the specific tradeoff between fitting parameters to a probabilistic model, versus direct search for a good classifier. Specifically, they illustrate the dangers of predicting using a model that is even "slightly" simpler than the distribution actually generating the data, vs. the relative robustness of directly searching for a good predictor. This would seem to be an important practical issue, and highlights the need for some better theoretical understanding of the notion of "robustness".

## 2 Conditionally Independent Attributes

Throughout we assume that each example is a binary vector $\tilde{x} \in \{0, 1\}^{n+1}$, and that each example is generated independently at random according to some unknown distribution on $\{0, 1\}^{n+1}$. We use $X_i$ to denote the $i$'th attribute, considered as a random variable, and $x_i$ to denote a value for $X_i$. In the conditionally independent attribute (CIA) model, examples are generated as follows. We postulate a "hidden" variable $Z$ with $k$ values, which takes values $z$ for $0 \le z < k$ with probability $\alpha_z \ge 0$. Since we must have $\sum_{z=0}^{k-1} \alpha_z = 1$ there are $k - 1$ independent parameters. Having randomly chosen a value $z$ for the hidden variable, we choose the value $x_i$ for each observable $X_i$: the value is 1 with probability $p_i^{(z)}$, and 0 otherwise. Here $p_i^{(z)} \in [0, 1]$. The attributes' values are chosen independently of each other, although $z$ remains fixed. Note that there are thus $(n + 1)k$ probability parameters $p_i^{(z)}$. In the following, let $\mathcal{P}$ denote the set of all $(n + 1)k + k - 1$ parameters in the model. From this point, and until Section 7, we always assume that $k = 2$ and in this case, to simplify notation, we write $\alpha_1$ as $\alpha$, $\alpha_0$ $(= 1 - \alpha)$ as $\alpha'$, $p_i^1$ as $p_i$ and $p_i^0$ as $q_i$.

## 3    The Expectation-Maximization algorithm (EM)

One traditional unsupervised approach to learning the parameters of this model is to find the maximum-likelihood parameters of the distribution given the data. That is, we attempt to find the set of parameters that maximizes the probability of the data observed.

Finding the maximum likelihood parameterization analytically appears to be a difficult problem, even in this rather simple setting. However, a practical approach is to use the well-known *Expectation-Maximization* algorithm (EM) [DLR77], which is an iterative approach that always converges to a local maximum of the likelihood function. In our setting, the procedure is as follows. We simply begin with a randomly chosen parameterization $\mathcal{P}$, and then we iterate until (apparent) convergence:[1]

**Expectation**: For all $\tilde{x}^i$, compute $u_i = P_{\mathcal{P}}(\tilde{x}^i \wedge z = 1)$ and $v_i = P_{\mathcal{P}}(\tilde{x}^i \wedge z = 0)$.

**Maximization**: Reestimate $\mathcal{P}$ as follows (writing $U = \sum_i u_i$ and $V = \sum_i v_i$):

$$\alpha \leftarrow \textstyle\sum_{i=1}^{\ell} u_i/(U+V) \qquad p_j \leftarrow \sum_{\{i:\tilde{x}_j^i=1\}} u_i/U \qquad q_j \leftarrow \sum_{\{i:\tilde{x}_j^i=1\}} v_i/V.$$

After convergence has been detected all we know is that we are near a *local* minima of the likelihood function. Thus it is prudent to repeat the process with many different restarts. (All our experiments were extremely conservative concerning the stopping criteria at each iteration, and in the number of iterations we tried.) But in practice, we are never sure that the true optimum has been located.

## 4    Covariances-Based approach

Partly in response to concern just expressed, we also developed another heuristic technique for learning $\mathcal{P}$. The algorithm, which we call COV, is based on measuring the covariance between pairs of attributes. Since we do not see $Z$, attributes will appear to be correlated. In fact, if the CIA model is correct, it is easy to show that covariance between $X_i$ and $X_j$ (defined as $y_{i,j} = \mu_{i,j} - \mu_i\mu_j$ where $\mu_i, \mu_j, \mu_{i,j}$ are the expectations of $X_i, X_j, (X_i \text{ and } X_j)$, respectively), will be $y_{i,j} = \alpha\alpha'\delta_i\delta_j$ where $\delta_i$ denotes $p_i - q_i$. We also know that the expected value of $X_i$ is $\mu_i = \alpha p_i + \alpha'q_i$. Furthermore, we will be able to get very accurate estimates of $\mu_i$ just by observing the proportion of samples in which $x_i$ is 1. Thus, if we could estimate both $\alpha$ and $\delta_i$ it would be trivial to solve for estimates of $p_i$ and $q_i$.

To estimate $\delta_i$, suppose we have computed all the pairwise covariances using the data; we use $\hat{y}_{i,j}$ to denote our estimate of $y_{i,j}$. For any distinct $j, k \neq i$ we clearly have $\alpha\alpha'\delta_i^2 = |\frac{y_{i,j}y_{i,k}}{y_{j,k}}|$ so we could estimate $\delta_i^2$ using this equation. A better estimate would be to consider *all* pairs $j, k$ and average the individual estimates. However, not all individual estimates are equally good. It can be shown that the smaller $y_{j,k}$ is, the less reliable we should expect the estimate to be (and in the limit, where $X_j$ and $X_k$ are perfectly uncorrelated, we get no valid estimate at all). This suggests that we use a weighted average, with the weights proportional to $y_{j,k}$. Using these weights leads us to the next equation for determining $\delta_i$, which, after simplification, is:

$$\alpha\alpha'\delta_i^2 \;=\; \frac{\sum_{j,k:j\neq k\neq i}|y_{i,j}y_{i,k}|}{\sum_{j,k:j\neq k\neq i}|y_{j,k}|} \;=\; \frac{(\sum_{j:j\neq i}|y_{i,j}|)^2 - \sum_{j:j\neq i}y_{i,j}^2}{\sum_{j,k:j\neq k}|y_{j,k}| - 2\sum_{j:j\neq i}|y_{j,i}|}$$

By substituting the estimates $\hat{y}_{i,j}$ we get an estimate for $\alpha\alpha'\delta_i^2$. This estimate can be computed in linear time except for the determination of $\sum_{j,k:j\neq k}|y_{j,k}|$ which, although quadratic, does not depend on $i$ and so can be computed once and for all. Thus it takes $O(n^2)$ time in total to estimate $\alpha\alpha'\delta_i^2$ for all $i$.

It remains only to estimate $\alpha$ and the signs of the $\delta_i$'s. Briefly, to determine the signs we first stipulate that $\delta_0$ is positive. (Because we never see $z$, one sign can be chosen at random.)

In principle, then, the sign of $\delta_j$ will then be equal to the sign of $y_{0,j}$, which we have an estimate for. In practice, this can statistically unreliable for small sample sizes and so we use a more involved "voting" procedure (details omitted here). Finally we estimate $\alpha$. We have found no better method of doing this than to simply search for the optimal value, using likelihood as the search criterion. However, this is only a 1-dimensional search and it turns out to be quite efficient in practice.

## 5   Linear Separators and CIA

Given a fully parameterized CIA model, we may be interested in predicting the value of one variable, say $X_0$, given known values for the remaining variables. One can show that in fact the optimal prediction region is given by a linear separator in the other variables, although we omit details of this derivation here.[2]   This suggest an obvious learning strategy: simply try to find the line which minimizes this loss on the training set. Unfortunately, in general the task of finding a linear separator that minimizes disagreements on a collection of examples is known to be NP-hard [HS92]. So instead we use an algorithm called *Winnow* that is known to produce good results when a linear separator exists, as well as under certain more relaxed assumptions [Lit91], and appears to be quite effective in practice.

## 6   Learning using a Winnow-based algorithm

The basic version of the Winnow algorithm [Lit88] keeps an $n$-dimensional vector $w = (w_1, \ldots w_n)$ of positive weights (i.e., $w_i$ is the weight associated with the $i$th feature), which it updates whenever a mistake is made. Initially, the weight vector is typically set to assign equal positive weight to all features. The algorithm has 3 parameters, a promotion parameter $\alpha > 1$, a demotion parameter $0 < \beta < 1$ and a threshold $\theta$. For a given instance $(x_1, \ldots, x_n)$ the algorithm predicts that $x_0 = 1$ iff $\sum_{i=1}^{m} w_i x_i > \theta$. If the algorithm predicts 0 and the label (i.e., $x_0$) is 1 (positive example) then the weights which correspond to active attributes ($x_i = 1$) are promoted—the weight $w_i$ is replaced by a larger weight $\alpha \cdot w_i$. Conversely, if algorithm predicts 1 and the received label is 0, then the weights which correspond to active features are demoted by factor $\beta$. We allow for negative weights as follows. Given an example $(x_1, \ldots, x_n)$, we rewrite it as an example over $2n$ variables $(y_1, y_2, \ldots, y_{2n})$ where $y_i = x_i$ and $y_{n+i} = 1 - x_i$. We then apply Winnow just as above to learn $2n$ (positive) weights. If $w_i^+$ is the weight associated with $x_i$ and $w_i^-$ is the weight associated with $x_{n+i}$ (i.e., $1 - x_i$), then the prediction rule is simply to compare $\sum_{i=1}^{n}(w_i^+ x_i + w_i^-(1 - x_i))$ with the threshold.

In the experiments described here we have made two significant modifications to the basic algorithm. To reduce variance, our final classifier is a weighted average of several classifiers; each is trained using a subsample from the training set, and its weight is based based on how well it was doing on that sample. Second, we biased the algorithm so as to look for "thick" classifiers. To understand this, consider the case in which the data is perfectly linearly separable. Then there will generally be many linear concepts that separate the training data we actually see. Among these, it seems plausible that we have a better chance of doing well on the unseen test data if we choose a linear concept that separates the positive and negative training examples as "widely" as possible. The idea of having a wide separation is less clear when there is no perfect separator, but we can still appeal to the basic intuition. To bias the search towards "thick" separators, we change Winnow's training rule somewhat. We now have a new margin parameter $\tau$. As before, we always update when our current hypothesis makes a mistake, but now we *also* update if $|\sum_{i=1}^{n} w_i x_i - \theta|$ is less than $\tau$, even if the prediction is correct. In our experiments, we found that performance when using this version of Winnow is better than that of the basic algorithm, so in this paper we present results for the former.

## 7  Experimental Methodology

Aside from the choice of algorithm used, the number of attributes $n$, and the sample size $s$, our experiments also differed in two other dimensions. These are the type of process generating the data (we will be interested in various deviations from CIA), and the "difficulty" of the problem. These features are determined by the *data model* we use (i.e., the distribution over $\{0, 1\}^n$ used to generate data sets).

Our first experiments consider the case where the data really is drawn from a binary CIA distribution. We associated with any such distribution a "difficulty" parameter $B$, which is the accuracy with which one could predict the value of $Z$ if one actually knew the correct model. (Of course, even with knowledge of the correct model we should not expect 100% accuracy.) The ability to control $B$ allows us to select and study models with different qualitative characteristics. In particular, this has allowed us concentrated most of our experiments on fairly "hard" instances[3], and to more meaningfully compare trials with differing numbers of attributes. We denote by $CIA(n, 2, b)$ the class of all data models which are binary CIA distributions over $n$ variables with difficulty $b$.[4] The next family of data models we used are also CIA models, but now using more than two values for the hidden variable. We denote the family using $k$ values as $CIA(n, k, b)$ where $n$ and $b$ are as before. When $k > 2$ there are more complex correlation patterns between the $X_i$ than when $k = 2$. Furthermore, the optimal predictor is not necessarily linear. The specific results we discuss in the next section have concentrated on this case.

Given any set of parameters, including a particular class of data models, our experiments are designed with the goal of good statistical accuracy. We repeatedly (typically 100 to 300 times) choose a data model at random from the chosen class, choose a sample of the appropriate size from this model, and then run all our algorithms. Each algorithm produces a (linear) hypothesis. We measure the success rate $S_{alg}$ (i.e., the proportion of times a hypothesis makes the correct prediction of $x_0$) by drawing yet more random samples from the data model being used. In the test phase we always draw enough new samples so that the confidence interval for $S_{alg}$, *for the results on a single model*, has width at most $\pm 1\%$. We use the $S_{alg}$ values to construct a *normalized* measure of performance (denoted $T$) as follows. Let $S_{best}$ be the best possible accuracy attainable for predicting $x_0$ (i.e., the accuracy achieved by the actual model generating the data). Let $S_{const}$ denote the performance of the best possible constant prediction rule (i.e., the rule that predicts the most likely *a priori* value for $x_0$). Note that $S_{const}$ and $S_{best}$ can vary from model to model. For each model we compute $\frac{S_{alg} - S_{const}}{S_{best} - S_{const}}$, and our normalized statistic $T$ is the average of these values. It can be thought of as measuring the percentage of the *possible* predictive power, over a plausible baseline, that an algorithm achieves.

## 8  Results

We only report on a small, but representative, selection of our experiments in any detail. For instance, although we have considered many values of $n$ ranging from 10 to 500, here we show six graphs giving the learning curves for $CIA(n, k, 0.90)$ for $n = 10, 75$, and for $k = 2, 3, 5$; as noted, we display the $T$ statistic. The error bars show the standard error,[5] providing a rough indication of accuracy. Not surprisingly, when the data model is binary

CIA, the EM algorithm does extremely well, learning significantly (if not overwhelmingly) faster than Winnow. But as we depart from the binary CIA assumption, the performance of EM quickly degrades.

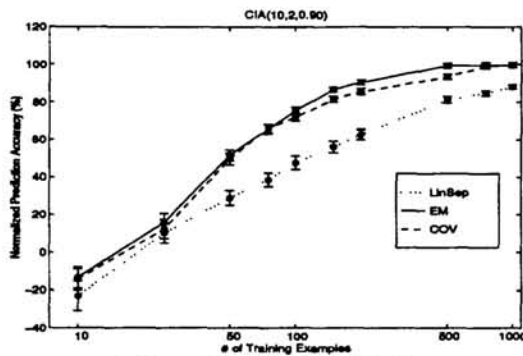

Figure 1: CIA(10,2,0.9)

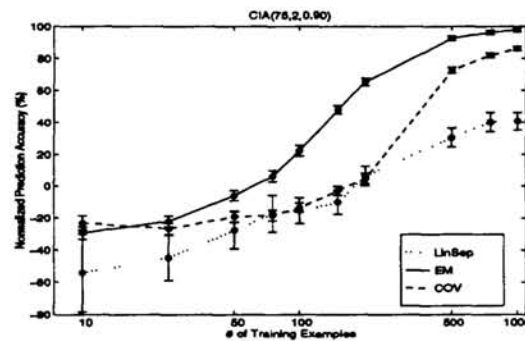

Figure 2: CIA(75,2,0.9)

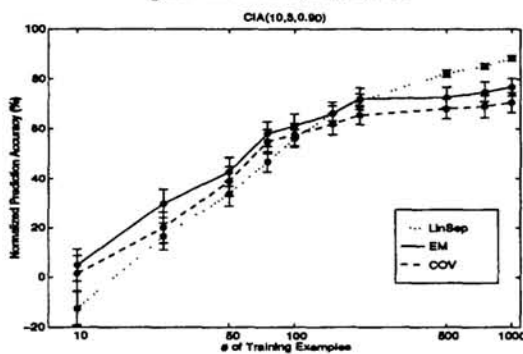

Figure 3: CIA(10,3,0.9)

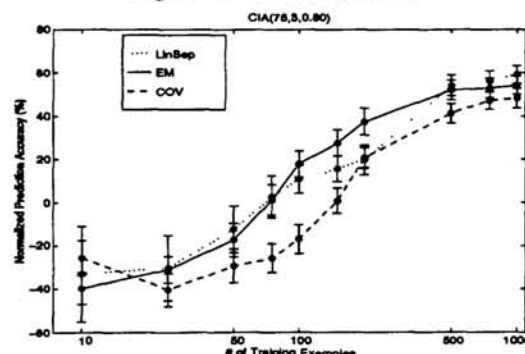

Figure 4: CIA(75,3,0.9)

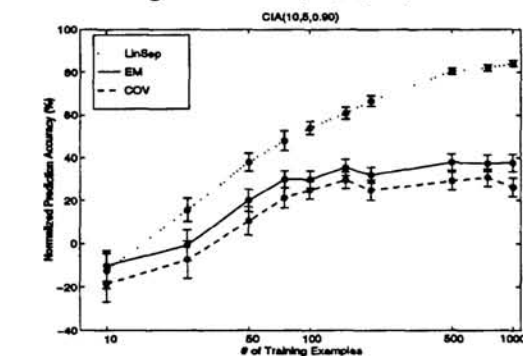

Figure 5: CIA(10,5,0.9)

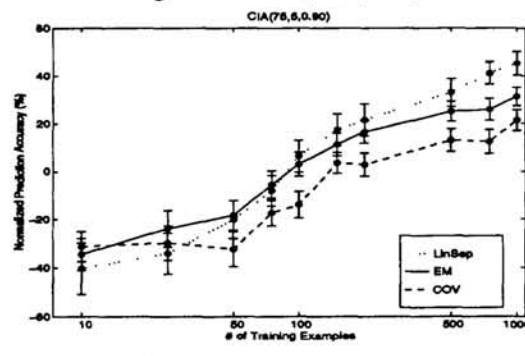

Figure 6: CIA(75,5,0.9)

When $k = 3$ performances is, on the whole, very similar for Winnow and EM. But when $k = 5$ Winnow is already superior to EM; significantly and uniformly so for $n = 10$. For fixed $k$ the difference seems to become somewhat less dramatic as $n$ increases; in Figure 6 (for $n = 75$) Winnow is less obviously dominant, and in fact is not better than EM until the sample size has reached 100. (But when $s \leq n$, meaning that we have fewer samples than attributes, the performance is uniformly dismal anyway.)

Should we attribute this degradation to the binary CIA assumption, or to the EM itself? This question is our reason for also considering the covariance algorithm. We see that the results for COV are generally similar to EM's, supporting our belief that the phenomena we see are properties inherent to the model rather than to the specific algorithm being used. Similarly (the results are omitted) we have tried several other algorithms that try to find good linear separators directly, including the classic *Perceptron* algorithm [MP69]; our version of Winnow was the best on the experiments we tried and thus we conjecture that its performance is (somewhat) indicative of what is possible for any such approach.

As the comparison between $n = 10$ and $n = 75$ illustrates, there is little qualitative differ-

ence between the phenomena observed as the number of attributes increases. Nevertheless, as $n$ grows it does seem that Winnow needs more examples before its performance surpasses that of the other algorithms (for any fixed $k$). As already noted, this may be due simply to the very "noisy" nature of the region $s \leq n$. We also have reasons to believe that this is partially an artifact of way we select models.

As previously noted, we also experimented with varying "difficulty" ($B$) levels. Although we omit the corresponding figures we mentioned that the main difference is that Winnow is a little faster in surpassing EM when the data deviates from the assumed model, but when the data model really is binary CIA, and EM converge even faster to an optimal performance.

These patterns were confirmed when we tried to compare the approaches on real data. We have used data that originates from a problem in which assuming a hidden "context" variable seems somewhat plausible. The data is taken from the context-sensitive spelling correction domain. We used one data set from those that were used in [GR96]. For example, given sentences in which the word *passed* or *past* appear, the task is to determine, for each such occurrence, which of the two it should be. This task may be modeled by thinking of the "context" as a hidden variable in our sense. Yet when we tried to learn in this case under the CIA model, with a binary valued hidden variable, the results were no better than just predicting the most likely classification (around 70%). Winnow, in contrast, performed extremely well and exceeds 95% on this task. We hesitate to read much into our limited real-data experiments, other than to note that so far they are consistent with the more careful experiments on synthetic data.

## 9  Conclusion

By restricting to a binary hidden variable, we have been able to consider a "fair" comparison between probabilistic model construction, and more traditional algorithms that directly learn a classification—at least in the sense that both have the same expressive power. Our conclusions concerning the fragility of the former should not be surprising but we believe that given the importance of the problem it is valuable to have some idea of the true significance of the effect. As we have indicated, in many real-world cases, where a model of the sort we have considered here seems plausible, it is impossible to nail down more specific characterizations of the probabilistic model. Our results exhibit how important it is to use the correct model and how sensitive are the results to deviations from it, when attempting to learn using model construction. The purpose of this paper is not to advocate that in practice one should use either Winnow or binary CIA in exactly the form considered here. A richer probabilistic model should be used along with a model selection phase. However, studying the problem in a restricted and controlled environment in crucial so as to understand the nature and significance of this fundamental problem.

## Footnotes

[1]The maximization phase works as though we were estimating parameters by taking averages based on weighted *labeled* data (i.e., in which we see $z$). If $\tilde{x}_i$ is a sample point, these fictional data points are $(\tilde{x}_i, z = 1)$ with weight $u_i/U$ and $(\tilde{x}_i, z = 0)$ with weight $v_i/V$.

[2]A derivation for the slightly different case, for predicting $z$, can be found in [MP69].

[3]Note that if one simply chooses parameters of a CIA model independently at random, without examining the difficulty of the model or adjusting for $n$, one will get many trivial problems, in which it is easy to predict $Z$ with nearly 100% accuracy, and thus predict optimally for $X_0$.

[4]It is nontrivial to efficiently select random models from this class. Briefly, our scheme is to choose each parameter in a CIA model independently from a symmetric beta distribution. Thus, the model parameters will have expected value 0.5. We choose the parameter of the beta distribution (which determines concentration about 0.5) so that the average $B$ value, of the models thus generated, equals $b$. Finally, we use rejection sampling to find CIA models with $B$ values that are exactly $b \pm 1\%$.

[5]Computed as the observed standard deviation, divided by the square root of the number of trials.

### References

[Blu97] A. Blum. Empirical support for winnow and weighted majority based algorithms: results on a calendar scheduling domain. *Machine Learning*, 26:1–19, 1997.

[DLR77] A. P. Dempster, N. M. Laird, and D. B. Rubin. Maximum likelihood from incomplete data via the EM algorithm. *Royal Statistical Society B*, 39:1–38, 1977.

[GR96] A. R. Golding and D. Roth. Applying winnow to context-sensitive spelling correcton. In *Proc. 13th International Conference on Machine Learning (ML'96)*, pages 182–190, 1996.

[HS92] K. Höffgen and H. Simon. Robust trainability of single neurons. In *Proc. 5th Annu. Workshop on Comput. Learning Theory*, pages 428–439, New York, New York, 1992. ACM Press.

[Lit88] N. Littlestone. Learning quickly when irrelevant attributes abound: A new linear-threshold algorithm. *Machine Learning*, 2:285–318, 1988.

[Lit91] N. Littlestone. Redundant noisy attributes, attribute errors, and linear threshold learning using Winnow. In *Proc. 4th Annu. Workshop on Comput. Learning Theory*, pages 147–156, San Mateo, CA, 1991. Morgan Kaufmann.

[MP69] M. L. Minsky and S. A. Papert. *Perceptrons*. MIT Press, Cambridge, MA, 1969.
